# Regularisation in Sequential Learning Algorithms

**João FG de Freitas**
Cambridge University
Engineering Department
Cambridge CB2 1PZ England
jfgf@eng.cam.ac.uk
[**Corresponding author**]

**Mahesan Niranjan**
Cambridge University
Engineering Department
Cambridge CB2 1PZ England
niranjan@eng.cam.ac.uk

**Andrew H Gee**
Cambridge University
Engineering Department
Cambridge CB2 1PZ England
ahg@eng.cam.ac.uk

## Abstract

In this paper, we discuss regularisation in online/sequential learning algorithms. In environments where data arrives sequentially, techniques such as cross-validation to achieve regularisation or model selection are not possible. Further, bootstrapping to determine a confidence level is not practical. To surmount these problems, a minimum variance estimation approach that makes use of the extended Kalman algorithm for training multi-layer perceptrons is employed. The novel contribution of this paper is to show the theoretical links between extended Kalman filtering, Sutton's variable learning rate algorithms and Mackay's Bayesian estimation framework. In doing so, we propose algorithms to overcome the need for heuristic choices of the initial conditions and noise covariance matrices in the Kalman approach.

## 1  INTRODUCTION

Model estimation involves building mathematical representations of physical processes using measured data. This problem is often referred to as system identification, time-series modelling or machine learning. In many occasions, the system being modelled varies with time. Under this circumstance, the estimator needs to be

updated sequentially. Online or sequential learning has many applications in tracking and surveillance, control systems, fault detection, communications, econometric systems, operations research, navigation and other areas where data sequences are often non-stationary and difficult to obtain before the actual estimation process.

To achieve acceptable generalisation, the complexity of the estimator needs to be judiciously controlled. Although there are various reliable schemes for controlling model complexity when training *en bloc* (batch processing), the same cannot be said about sequential learning. Conventional regularisation techniques cannot be applied simply because there is no data to cross-validate. Consequently, there is ample scope for the design of sequential methods of controlling model complexity.

## 2 NONLINEAR ESTIMATION

A dynamical system may be described by the following discrete, stochastic state space representation:

$$\mathbf{w}_{k+1} = \mathbf{w}_k + \mathbf{d}_k \tag{1}$$

$$y_k = \mathbf{g}(\mathbf{w}_k, \mathbf{t}_k) + \mathbf{v}_k \tag{2}$$

where it has been assumed that the model parameters $(\mathbf{w}_k \in \Re^q)$ constitute the states of the system, which in our case represent the weights of a multi-layer perceptron (MLP). $\mathbf{g}$ is a nonlinear vector function that may change at each estimation step $k$, $t_k$ denotes the time at the $k$-th estimation step and $d_k$ and $v_k$ represent zero mean white noise with covariances given by $Q_k$ and $R_k$ respectively. The noise terms are often called the process noise $(d_k)$ and the measurement noise $(v_k)$. The system measurements are encoded in the output vector $y_k \in \Re^m$.

The estimation problem may be reformulated as having to compute an estimate $\hat{\mathbf{w}}_k$ of the states $\mathbf{w}_k$ using the set of measurements $Y_k = \{y_1, y_2, \cdots, y_k\}$. The estimate $\hat{\mathbf{w}}_k$ can be used to predict future values of the output $y$. We want $\hat{\mathbf{w}}_k$ to be an unbiased, minimum variance and consistent estimate (Gelb 1984). A minimum variance (unbiased) estimate is one that has its variance less than or equal to that of any other unbiased estimator. Since the variance of the output $y$ depends directly on the variance of the parameter estimates (Åström 1970), the minimum variance framework constitutes a regularisation scheme for sequential learning.

The conditional probability density function of $\mathbf{w}_k$ given $Y_k$ ($p(\mathbf{w}_k|Y_k)$) constitutes the complete solution of the estimation problem (Bar-Shalom and Li 1993, Ho and Lee 1964, Jazwinski 1970). This is simply because $p(\mathbf{w}_k|Y_k)$ embodies all the statistical information about $\mathbf{w}_k$ given the measurements $Y_k$ and the initial condition $\mathbf{w}_0$. This is essentially the Bayesian approach to estimation, where instead of describing a model by a single set of parameters, it is expressed in terms of the conditional probability $p(\mathbf{w}_k|Y_k)$ (Jaynes 1986, Jazwinski 1970). The estimate $\hat{\mathbf{w}}_k$ can be computed from $p(\mathbf{w}_k|Y_k)$ according to several criteria, namely MAP estimation (peak of the posterior), minimum variance estimation (centroid of the posterior) and minimax estimation (median of the posterior).

The Bayesian solution to the optimal estimation problem is (Ho and Lee 1964):

$$\begin{aligned} p(\mathbf{w}_{k+1}|Y_{k+1}) &= \frac{p(\mathbf{w}_{k+1}, y_{k+1}|Y_k)}{p(y_{k+1}|Y_k)} \\ &= \frac{\int p(y_{k+1}|Y_k, \mathbf{w}_{k+1})p(\mathbf{w}_{k+1}|\mathbf{w}_k)p(\mathbf{w}_k|Y_k)d\mathbf{w}_k}{\int \int p(y_{k+1}|Y_k, \mathbf{w}_{k+1})p(\mathbf{w}_{k+1}|\mathbf{w}_k)p(\mathbf{w}_k|Y_k)d\mathbf{w}_{k+1}d\mathbf{w}_k} \end{aligned} \tag{3}$$

where the integrals run over the parameter space. This functional integral difference equation governing the evolution of the posterior density function is not suitable

for practical implementation (Bar-Shalom and Li 1993, Jazwinski 1970). It involves propagating a quantity (the posterior density function) that cannot be described by a finite number of parameters. The situation in the linear case is vastly simpler. There the mean and covariance are sufficient statistics for describing the Gaussian posterior density function.

In view of the above statements, it would be desirable to have a framework for non-linear estimation similar to the one for linear-Gaussian estimation. The extended Kalman filter (EKF) constitutes an attempt in this direction (Bar-Shalom and Li 1993, Gelb 1984). The EKF is a minimum variance estimator based on a Taylor series expansion of the nonlinear function $\mathbf{g}(\mathbf{w})$ around the previous estimate. The EKF equations for a linear expansion are given by:

$$K_{k+1} = (P_k + Q_k)G_{k+1}[R_k + G_{k+1}^T(P_k + Q_k)G_{k+1}]^{-1} \tag{4}$$

$$\hat{\mathbf{w}}_{k+1} = \hat{\mathbf{w}}_k + K_{k+1}(y_{k+1} - G_{k+1}^T\hat{\mathbf{w}}_k) \tag{5}$$

$$P_{k+1} = P_k + Q_k - K_{k+1}G_{k+1}^T(P_k + Q_k) \tag{6}$$

where $P_k$ denotes the covariance of the weights. In the general multiple input, multiple output (MIMO) case, $\mathbf{g} \in \Re^m$ is a vector function and $G$ represents the Jacobian of the network outputs with respect to the weights.

The EKF provides a minimum variance Gaussian approximation to the posterior probability density function. In many cases, $p(\mathbf{w}_k|Y_k)$ is a multi-modal (several peaks) function. In this scenario, it is possible to use a committee of Kalman filters, where each individual filter approximates a particular mode, to produce a more accurate approximation (Bar-Shalom and Li 1993, Kadirkamanathan and Kadirkamanathan 1995). The parameter covariances of the individual estimators may be used to determine the contribution of each estimator to the committee. In addition, the parameter covariances serve the purpose of placing confidence intervals on the output prediction.

## 3  TRAINING MLPs WITH THE EKF

One of the earliest implementations of EKF trained MLPs is due to Singhal and Wu (Singhal and Wu 1988). In their method, the network weights are grouped into a single vector $\mathbf{w}$ that is updated in accordance with the EKF equations. The entries of the Jacobian matrix are calculated by back-propagating the $m$ output values through the network.

The algorithm proposed by Singhal and Wu requires a considerable computational effort. The complexity is of the order $mq^2$ multiplications per estimation step. Shah, Palmieri and Datum (1992) and Puskorius and Feldkamp (1991) have proposed strategies for decoupling the global EKF estimation algorithm into local EKF estimation sub-problems, thereby reducing the computational time. The EKF is an improvement over conventional MLP estimation techniques, such as back-propagation, in that it makes use of second order statistics (covariances). These statistics are essential for placing error bars on the predictions and for combining separate networks into committees of networks. Further, it has been proven elsewhere that the back-propagation algorithm is simply a degenerate of the EKF algorithm (Ruck, Rogers, Kabrisky, Maybeck and Oxley 1992).

However, the EKF algorithm for training MLPs suffers from serious difficulties, namely choosing the initial conditions $(\mathbf{w}_0, P_0)$ and the noise covariance matrices $R$ and $Q$. In this work, we propose the use of maximum likelihood techniques, such as back-propagation computed over a small set of initial data, to initialise the

EKF-MLP estimator. The following two subsections· describe ways of overcoming the difficulty of choosing $R$ and $Q$.

## 3.1 ELIMINATING $Q$ BY UPDATING $P$ WITH BACK-PROPAGATION

To circumvent the problem of choosing the process noise covariance $Q$, while at the same time increasing computational efficiency, it is possible to extend an algorithm proposed by Sutton (Sutton 1992) to the nonlinear case. In doing so, the weights covariance is approximated by a diagonal matrix with entries given by $p_{qq} = \exp(\beta_q)$, where $\beta$ is updated by error back-propagation (de Freitas, Niranjan and Gee 1997).

The Kalman gain $K_k$ and the weights estimate $\hat{\mathbf{w}}_k$ are updated using a variation of the Kalman equations, where the Kalman gain and weights update equations are independent of $Q$ (Gelb 1984), while the weights covariance $P$ is updated by back-propagation. This algorithm lessens the burden of choosing the matrix $Q$ by only having to choose the learning rate scalar $\eta$. The performance of the EKF algorithm with $P$ updated by back-propagation will be analysed in Section 4.

## 3.2 KALMAN FILTERING AND BAYESIAN TECHNIQUES

A further improvement on the EKF algorithm for training MLPs would be to update $R$ and $Q$ automatically each estimation step. This can be done by borrowing some ideas from the Bayesian estimation field. In particular, we shall attempt to link Mackay's work (Mackay 1992, Mackay 1994) on Bayesian estimation for neural networks with the EKF estimation framework. This theoretical link should serve to enhance both methods.

Mackay expresses the prior, likelihood and posterior density functions in terms of the following Gaussian approximations:

$$p(\mathbf{w}) = \frac{1}{(2\pi)^{q/2}\alpha^{-q/2}} \exp\left( - \frac{\alpha}{2}\|\mathbf{w}\|^2 \right) \tag{7}$$

$$p(Y_k|\mathbf{w}) = \frac{1}{(2\pi)^{n/2}\beta^{-n/2}} \exp\left( - \frac{\beta}{2}\sum_{k=1}^{n}(y_k - \hat{f}_{n,q}(\mathbf{w}, \Phi_k))^2 \right) \tag{8}$$

$$p(\mathbf{w}|Y_k) = \frac{1}{(2\pi)^{q/2}|A|^{-1/2}} \exp\left( - \frac{1}{2}(\mathbf{w} - \mathbf{w}_{MP})^T A(\mathbf{w} - \mathbf{w}_{MP}) \right) \tag{9}$$

where $\hat{f}_{n,q}(\mathbf{w}, \Phi_k)$ represents the estimator and the hyper-parameters $\alpha$ and $\beta$ control the variance of the prior distribution of weights and the variance of the measurement noise. $\alpha$ also plays the role of the regularisation coefficient. The posterior is obtained by approximating it with a Gaussian function, whose mean $\mathbf{w}_{MP}$ is given by a minimum of the following regularised error function:

$$S(\mathbf{w}) = \frac{\alpha}{2}\|\mathbf{w}\|^2 + \frac{\beta}{2}\sum_{k=1}^{n}(y_k - \hat{f}_{n,q}(\mathbf{w}, \Phi_k))^2 \tag{10}$$

The posterior covariance $A$ is the Hessian of the above error function.

In Mackay's estimation framework, also known as the evidence framework, the parameters $\mathbf{w}$ are obtained by minimising equation (10), while the hyper-parameters $\alpha$ and $\beta$ are obtained by maximising the evidence $p(Y_k|\alpha, \beta)$ after approximating the posterior density function by a Gaussian function. In doing so, the following recursive formulas for $\alpha$ and $\beta$ are obtained:

$$\alpha_{k+1} = \frac{\gamma}{\sum_{i=1}^{q} w_i^2} \qquad \text{and} \qquad \beta_{k+1} = \frac{n - \gamma}{\sum_{k=1}^{n}(y_k - \hat{f}_{n,q}(\mathbf{w}_k, \Phi_k))^2}$$

The quantity $\gamma$ represents the effective number of parameters $\gamma = \sum_{i=1}^{q} \frac{\lambda_i}{\lambda_i + \alpha}$, where the $\lambda_i$ correspond to the eigenvalues of the Hessian of the error function without the regularisation term.

Instead of adopting Mackay's evidence framework, it is possible to maximise the posterior density function by performing integrations over the hyper-parameters analytically (Buntine and Weigend 1991, Mackay 1994). The latter approach is known as the MAP framework for $\alpha$ and $\beta$. The hyper-parameters computed by the MAP framework differ from the ones computed by the evidence framework in that the former makes use of the total number of parameters and not only the effective number of parameters. That is, $\alpha$ and $\beta$ are updated according to:

$$\alpha_{k+1} = \frac{q}{\sum_{i=1}^{q} w_i^2} \qquad \text{and} \qquad \beta_{k+1} = \frac{n}{\sum_{k=1}^{n} (y_k - \hat{f}_{n,q}(\mathbf{w}_k, \Phi_k))^2}$$

By comparing the equations for the prior, likelihood and posterior density functions in the Kalman filtering framework (Ho and Lee 1964) with equations (7), (8) and (9) we can establish the following relations:

$$P = A^{-1} \ , \qquad\qquad Q = \alpha^{-1} I_q - A^{-1} \qquad \text{and} \qquad R = \beta^{-1} I_m$$

where $I_q$ and $I_m$ represent identity matrices of sizes $q$ and $m$ respectively. Therefore, it is possible to update $Q$ and $R$ sequentially by expressing them in terms of the sequential updates of $\alpha$ and $\beta$.

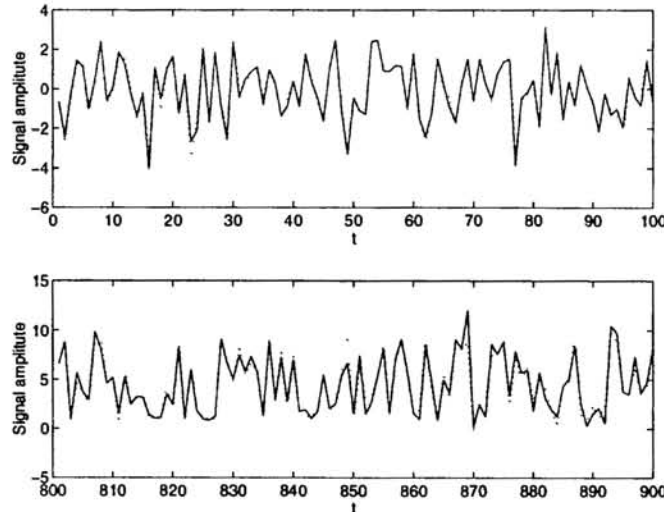

Figure 1: Prediction using the conventional EKF algorithm for a network with 20 hidden neurons. Actual output [$\cdots$] and estimated output [—].

## 4   RESULTS

To compare the performance of the conventional EKF algorithm, the EKF algorithm with $P$ updated by back-propagation, and the EKF algorithm with $R$ and $Q$ updated sequentially according to the Bayesian MAP framework, noisy data was generated from the following nonlinear, non-stationary, multivariate process:

$$y(t) = \begin{cases} x_1(t) + x_2(t) + v(t) & 1 \le t \le 200 \\ 4\sin(x_1(t)) + x_2(t)\sin(0.03(t - 200)) + v(t) & 200 < t \le 1000 \end{cases}$$

where the inputs $x_i$ are uniformly distributed random sequences with variance equal · to 1 and $v(t)$ corresponds to uniformly distributed noise with variance equal to 0.1. Figure 1 shows the prediction obtained using the conventional EKF algorithm. To

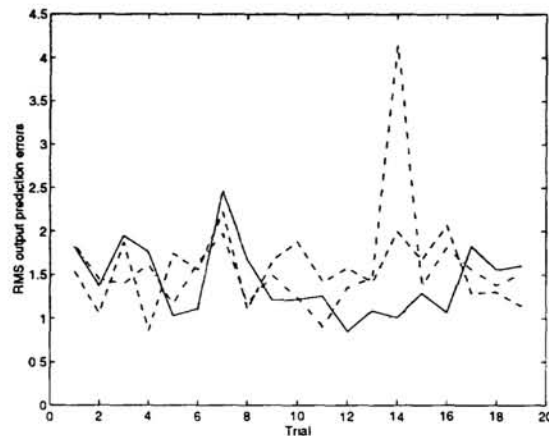

Figure 2: Output error for the conventional EKF algorithm [· · ·], the EKF algorithm with $P$ updated by back-propagation [- . -], the EKF algorithm with $R$ and $Q$ updated sequentially according to the Bayesian MAP framework [—], and the EKF algorithm with the Bayesian evidence framework [- - -].

compare the four estimation frameworks, an MLP with 20 neurons in the hidden layer was selected. The initial conditions were obtained by using back-propagation on the first 100 samples and assigning to $P$ a diagonal matrix with diagonal elements equal to 10. The matrices $R$ and $Q$ in the conventional EKF algorithm were chosen, by trial and error, to be identity matrices. In the EKF algorithm with P updated by back-propagation, $R$ was chosen to be equal to the identity matrix, while the learning rate was set to 0.01. Finally, in the EKF algorithm with $R$ and $Q$ updated sequentially, the initial $R$ and $Q$ matrices were chosen to be identity matrices. The prediction errors obtained for each method with random input data are shown in Figure 2.

It is difficult to make a fair comparison between the four nonlinear estimation methods because their parameters were optimised independently. However, the results suggest that the prediction obtained with the conventional EKF training outperforms the predictions of the other methods. This may be attributed to the facts that, firstly, in this simple problem it is possible to guess the optimal values for $R$ and $Q$ and, secondly, the algorithms to update the noise covariances may affect the regularisation performance of the EKF algorithm. This issue, and possible solutions, is explored in depth by the authors in (de Freitas et al. 1997).

## 5   Conclusions

In this paper, we point out the links between Kalman filtering, gradient descent algorithms with variable learning rates and Bayesian estimation. This results in two algorithms for eliminating the problem of choosing the initial conditions and the noise covariance matrices in the training of MLPs with the EKF. These algorithms are illustrated on a toy problem here, but more extensive experiments have been reported in (de Freitas et al. 1997).

Improved estimates may be readily obtained by combining the estimators into com-

mittees or extending the training methods to recurrent networks. Finally, the computational time may be reduced by decoupling the network weights.

## Acknowledgements

João FG de Freitas is financially supported by two University of the Witwatersrand Merit Scholarships, a Foundation for Research Development Scholarship (South Africa) and a Trinity College External Studentship (Cambridge).

## References

Åström, K. J. (1970). *Introduction to Stochastic Control Theory*, Academic Press.

Bar-Shalom, Y. and Li, X. R. (1993). *Estimation and Tracking: Principles, Techniques and Software*, Artech House, Boston.

Buntine, W. L. and Weigend, A. S. (1991). Bayesian back-propagation, *Complex Systems* **5**: 603–643.

de Freitas, J., Niranjan, M. and Gee, A. (1997). Hierarchichal Bayesian-Kalman models for regularisation and ARD in sequential learning, *Technical Report CUED/F-INFENG/TR 307*, Cambridge University, http://svr-www.eng.cam.ac.uk/˜jfgf.

Gelb, A. (ed.) (1984). *Applied Optimal Estimation*, MIT Press.

Ho, Y. C. and Lee, R. C. K. (1964). A Bayesian approach to problems in stochastic estimation and control, *IEEE Transactions on Automatic Control* **AC-9**: 333–339.

Jaynes, E. T. (1986). Bayesian methods: General background, *in* J. H. Justice (ed.), *Maximum Entropy and Bayesian Methods in Applied Statistics*, Cambridge University Press, pp. 1–25.

Jazwinski, A. H. (1970). *Stochastic Processes and Filtering Theory*, Academic Press.

Kadirkamanathan, V. and Kadirkamanathan, M. (1995). Recursive estimation of dynamic modular RBF networks, *in* D. S. Touretzky, M. C. Mozer and M. E. Hasselmo (eds), *Advances in Neural Information Processing Systems 8*, pp. 239–245.

Mackay, D. J. C. (1992). Bayesian interpolation, *Neural Computation* **4**(3): 415–447.

Mackay, D. J. C. (1994). Hyperparameters: Optimise or integrate out?, *in* G. Heidbreder (ed.), *Maximum Entropy and Bayesian Methods*.

Puskorius, G. V. and Feldkamp, L. A. (1991). Decoupled extended Kalman filter training of feedforward layered networks, *International Joint Conference on Neural Networks*, Seattle, pp. 307–312.

Ruck, D. W., Rogers, S. K., Kabrisky, M., Maybeck, P. S. and Oxley, M. E. (1992). Comparative analysis of backpropagation and the extended Kalman filter for training multilayer perceptrons, *IEEE Transactions on Pattern Analysis and Machine Intelligence* **14**(6): 686–690.

Shah, S., Palmieri, F. and Datum, M. (1992). Optimal filtering algorithms for fast learning in feedforward neural networks, *Neural Networks* **5**: 779–787.

Singhal, S. and Wu, L. (1988). Training multilayer perceptrons with the extended Kalman algorithm, *in* D. S. Touretzky (ed.), *Advances in Neural Information Processing Systems*, Vol. 1, San Mateo, CA, pp. 133–140.

Sutton, R. S. (1992). Gain adaptation beats least squares?, *Proceedings of the Seventh Yale Workshop on Adaptive Learning Systems*, pp. 161–166.